# A Gradient-Based Boosting Algorithm for Regression Problems

**Richard S. Zemel**      **Toniann Pitassi**
Department of Computer Science
University of Toronto

## Abstract

In adaptive boosting, several weak learners trained sequentially are combined to boost the overall algorithm performance. Recently adaptive boosting methods for classification problems have been derived as gradient descent algorithms. This formulation justifies key elements and parameters in the methods, all chosen to optimize a single common objective function. We propose an analogous formulation for adaptive boosting of regression problems, utilizing a novel objective function that leads to a simple boosting algorithm. We prove that this method reduces training error, and compare its performance to other regression methods.

The aim of boosting algorithms is to "boost" the small advantage that a *hypothesis* produced by a *weak* learner can achieve over random guessing, by using the weak learning procedure several times on a sequence of carefully constructed distributions. Boosting methods, notably AdaBoost (Freund & Schapire, 1997), are simple yet powerful algorithms that are easy to implement and yield excellent results in practice. Two crucial elements of boosting algorithms are the way in which a new distribution is constructed for the learning procedure to produce the next hypothesis in the sequence, and the way in which hypotheses are combined to produce a highly accurate output. Both of these involve a set of parameters, whose values appeared to be determined in an ad hoc manner. Recently boosting algorithms have been derived as gradient descent algorithms (Breiman, 1997; Schapire & Singer, 1998; Friedman et al., 1999; Mason et al., 1999). These formulations justify the parameter values as all serving to optimize a single common objective function.

These optimization formulations of boosting originally developed for classification problems have recently been applied to regression problems. However, key properties of these regression boosting methods deviate significantly from the classification boosting approach. We propose a new boosting algorithm for regression problems, also derived from a central objective function, which retains these properties.

In this paper, we describe the original boosting algorithm and summarize boosting methods for regression. We present our method and provide a simple proof that elucidates conditions under which convergence on training error can be guaranteed. We propose a probabilistic framework that clarifies the relationship between various optimization-based boosting methods. Finally, we summarize empirical comparisons between our method and others on some standard problems.

# 1 A Brief Summary of Boosting Methods

Adaptive boosting methods are simple modular algorithms that operate as follows. Let $g : \mathbf{X} \to Y$ be the function to be learned, where the label set $Y$ is finite, typically binary-valued. The algorithm uses a learning procedure, which has access to $n$ training examples, $\{(\mathbf{x}_1, y_1), \ldots, (\mathbf{x}_n, y_n)\}$, drawn randomly from $\mathbf{X} \times Y$ according to distribution $\mathcal{D}$; it outputs a hypothesis $f : \mathbf{X} \to Y$, whose error is the expected value of a loss function on $f(\mathbf{x}), g(\mathbf{x})$, where $\mathbf{x}$ is chosen according to $\mathcal{D}$. Given $\epsilon, \delta > 0$ and access to random examples, a *strong* learning procedure outputs with probability $1 - \delta$ a hypothesis with error at most $\epsilon$, with running time polynomial in $1/\epsilon$, $1/\delta$ and the number of examples. A *weak* learning procedure satisfies the same conditions, but where $\epsilon$ need only be better than random guessing.

Schapire (1990) showed that any weak learning procedure, denoted **WeakLearn**, can be efficiently transformed ("boosted") into a strong learning procedure. The AdaBoost algorithm achieves this by calling **WeakLearn** multiple times, in a sequence of $T$ stages, each time presenting it with a different distribution over a fixed training set and finally combining all of the hypotheses. The algorithm maintains a weight $w_t^i$ for each training example $i$ at stage $t$, and a distribution $\mathcal{D}_t$ is computed by normalizing these weights. The algorithm loops through these steps:

1. At stage $t$, the distribution $\mathcal{D}_t$ is given to **WeakLearn**, which generates a hypothesis $f_t$. The error rate $\epsilon_t$ of $f_t$ w.r.t. $\mathcal{D}_t$ is: $\epsilon_t = \sum_{i:f_t(\mathbf{x}^i) \neq y^i} w_t^i / \sum_{i=1}^{n} w_t^i$

2. The new training distribution is obtained from the new weights: $w_{t+1}^i = w_t^i * (\epsilon_t / (1 - \epsilon_t))^{1 - |f_t(\mathbf{x}^i) - y^i|}$

After $T$ stages, a test example $\mathbf{x}$ will be classified by a combined weighted-majority hypothesis: $\hat{y} = \mathbf{sgn}(\sum_{t=1}^{T} c_t f_t(\mathbf{x}))$. Each *combination coefficient* $c_t = \log((1 - \epsilon_t)/\epsilon_t)$ takes into account the accuracy of hypothesis $f_t$ with respect to its distribution.

The optimization approach derives these equations as all minimizing a common objective function $J$, the expected error of the combined hypotheses, estimated from the training set. The new hypothesis is the step in function space in the direction of steepest descent of this objective. For example, if $J = \frac{1}{n} \sum_{i=1}^{n} \exp(-\sum_t y^i c_t f_t(x^i))$, then the cost after $T$ rounds is the cost after $T - 1$ rounds times the cost of hypothesis $f_T$:

$$
\begin{aligned}
J(T) &= \frac{1}{n} \sum_{i=1}^{n} \exp(-\sum_{t=1}^{T-1} y^i c_t f_t(x^i)) \exp(-y^i c_T f_T(x^i)) \\
&= \sum_i w_T^i \exp(-y^i c_T f_T(x^i))
\end{aligned}
$$

so training $f_T$ to minimize $J(T)$ amounts to minimizing the cost on a weighted training distribution. Similarly, the training distribution is formed by normalizing updated weights: $w_{t+1}^i = w_t^i * \exp(-y^i c_t f_t(x^i)) = w_t^i * \exp(s_t^i c_t)$ where $s_t^i = 1$ if $f_t(x^i) \neq y^i$, else $s_t^i = -1$. Note that because the objective function $J$ is multiplicative in the costs of the hypotheses, a key property follows: The objective for each hypothesis is formed simply by re-weighting the training distribution.

This boosting algorithm applies to binary classification problems, but it does not readily generalize to regression problems. Intuitively, regression problems present special difficulties because hypotheses may not just be right or wrong, but can be a little wrong or very wrong. Recently a spate of clever optimization-based boosting methods have been proposed for regression (Duffy & Helmbold, 2000; Friedman,

1999; Karakoulas & Shawe-Taylor, 1999; Rätsch et al., 2000). While these methods involve diverse objectives and optimization approaches, they are alike in that new hypotheses are formed not by simply changing the example weights, but instead by modifying the target values. As such they can be viewed as forms of forward stage-wise additive models (Hastie & Tibshirani, 1990), which produce hypotheses sequentially to reduce *residual* error. We study a simple example of this approach, in which hypothesis $T$ is trained not to produce the target output $y^i$ on a given case $i$, but instead to fit the current residual, $r_T^i$, where $r_T^i = y^i - \sum_{t=1}^{T-1} c_t f_t(\mathbf{x})$. Note that this approach develops a series of hypotheses all based on optimizing a common objective, but it deviates from standard boosting in that the distribution of examples is not used to control the generation of hypotheses, and each hypothesis is not trained to learn the same function.

## 2   An Objective Function for Boosting Regression Problems

We derive a boosting algorithm for regression from a different objective function. This algorithm is similar to the original classification boosting method in that the objective is multiplicative in the hypotheses' costs, which means that the target outputs are not altered after each stage, but rather the objective for each hypothesis is formed simply by re-weighting the training distribution. The objective function is:

$$J_T = \frac{1}{n} \sum_{i=1}^{n} \left( \prod_{t=1}^{T} c_t^{-\frac{1}{2}} \right) \exp \left[ \sum_{t=1}^{T} c_t (f_t(x^i) - y^i)^2 \right] \tag{1}$$

Here, training hypothesis $T$ to minimize $J_T$, the cost after $T$ stages, amounts to minimizing the exponentiated squared error of a weighted training distribution:

$$\begin{aligned} J_T &= \frac{1}{n} \sum_{i=1}^{n} \left( (\prod_{t=1}^{T-1} c_t^{-\frac{1}{2}}) \exp \left[ \sum_{t=1}^{T-1} c_t (f_t(x^i) - y^i)^2 \right] \right) \left( c_T^{-\frac{1}{2}} \exp \left[ c_T (f_T(x^i) - y^i)^2 \right] \right) \\ &= \sum_{i=1}^{n} w_T^i \left( c_T^{-\frac{1}{2}} \exp \left[ c_T (f_T(x^i) - y^i)^2 \right] \right) \end{aligned}$$

We update each weight by multiplying by its respective error, and form the training distribution for the next hypothesis by normalizing these updated weights.

In the standard AdaBoost algorithm, the combination coefficient $c_t$ can be analytically determined by solving $\frac{\partial J_t}{\partial c_t} = 0$ for $c_t$. Unfortunately, one cannot analytically determine the combination coefficient $c_t$ in our algorithm, but a simple line search can be used to find value of $c_t$ that minimizes the cost $J_t$. We limit $c_t$ to be between 0 and 1. Finally, optimizing $J$ with respect to $y$ produces a simple linear combination rule for the estimate: $\hat{y} = \sum_t c_t f_t(\mathbf{x}) / \sum_t c_t$.

We introduce a constant $\tau$ as a threshold used to demarcate correct from incorrect responses. This threshold is the single parameter of this algorithm that must be chosen in a problem-dependent manner. It is used to judge when the performance of a new hypothesis warrants its inclusion: $\epsilon_t = \sum_i p_t^i \exp[(f_t(\mathbf{x}^i) - y^i)^2 - \tau] < 1$. The algorithm can be summarized as follows:

## 3  Proof of Convergence

**Theorem**: *Assume that for all $t \le T$, hypothesis $t$ makes error $\epsilon_t$ on its distribution. If the combined output $\hat{y}$ is considered to be in error iff $(\hat{y} - y)^2 > \tau$, then the output of the boosting algorithm (after $T$ stages) will have error at most $\epsilon$, where*

$$\epsilon = P[(\hat{y^i} - y^i)^2 > \tau] \le \prod_{t=1}^{T} \epsilon_t \exp[\tau(T - \sum_{t=1}^{T} c_t)]$$

.

**Proof:** We follow the approach used in the AdaBoost proof (Freund & Schapire, 1997). We show that the sum of the weights at stage $T$ is bounded above by a constant times the product of the $\epsilon_t$'s, while at the same time, for each input $i$ that is incorrect, its corresponding weight $w_T^i$ at stage $T$ is significant.

$$\sum_{i=1}^{n} w_{T+1}^i = \sum_i w_T^i c_T^{-1/2} \exp[c_T (f_T(\mathbf{x}^i) - y^i)^2] \le c_T^{-1/2} \epsilon_T \exp(\tau) \sum_i w_T^i$$

$$\le \prod_{t=1}^{T} c_T^{-1/2} \exp(\tau) \epsilon_t$$

The inequality holds because $0 \le c_t \le 1$. We now compute the new weights:

$$\sum_t c_t (f_t(\mathbf{x}^i) - y^i)^2 = [\sum_t c_t][\mathbf{Var}(f^i) + (\hat{y^i} - y^i)^2] \ge [\sum_t c_t][(\hat{y^i} - y^i)^2]$$

where $\hat{y^i} = \sum_t c_t f_t(\mathbf{x}^i) / \sum_t c_t$ and $\mathbf{Var}(f^i) = \sum_t c_t (f_t(\mathbf{x}^i) - \hat{y^i})^2 / \sum_t c_t$. Thus,

$$w_{T+1}^i = (\prod_{t=1}^{T} c_t^{-1/2}) \exp(\sum_{t=1}^{T} c_t (f_t(\mathbf{x}^i) - y^i)^2) \ge (\prod_{t=1}^{T} c_t^{-1/2}) \exp([\sum_{t=1}^{T} c_t][(\hat{y^i} - y^i)^2])$$

Now consider an example input $k$ such that the final answer is an error. Then, by definition, $(\hat{y}^k - y^k)^2 > \tau \Rightarrow w_{T+1}^k \geq (\prod_t c_t^{-1/2}) \exp(\tau \sum_t c_t)$. If $\epsilon$ is the total error rate of the combination output, then:

$$\sum_i w_{T+1}^i \geq \sum_{k:k\ \text{error}} w_{T+1}^k \geq \epsilon(\prod_{t=1}^{T} c_t^{-1/2}) \exp(\tau \sum_{t=1}^{T} c_t)$$

$$\epsilon \leq (\sum_i w_{T+1}^i)(\prod_{t=1}^{T} c_t^{1/2}) \exp[\tau(T - \sum_t c_t)] \leq \prod_{t=1}^{T} \epsilon_t \exp[\tau(T - \sum_t c_t)]$$

Note that as in the binary AdaBoost theorem, there are no assumptions made here about $\epsilon_t$, the error rate of individual hypotheses. If all $\epsilon_t = \Delta < 1$, then $\epsilon < \Delta^T \exp[\tau(T - \sum_t c_t)]$, which is exponentially decreasing as long as $c_t \to 1$.

## 4  Comparing the Objectives

We can compare the objectives by adopting a probabilistic framework. We associate a probability distribution with the output of each hypothesis on input $x$, and combine them to form a *consensus* model $M$ by multiplying the distributions: $g(y|x, M) \equiv \prod_t p_t(y|x, \theta_t)$, where $\theta_t$ are parameters specific to hypothesis $t$. If each hypothesis $t$ produces a single output $f_t(x)$ and has confidence $c_t$ assigned to it, then $p_t(y|x, \theta_t)$ can be considered a Gaussian with mean $f_t(x)$ and variance $1/c_t$

$$g(y|x, M) = k \left[\prod_t c_t^{1/2}\right] \exp\left[-\sum_t c_t(y - f_t(x))^2\right]$$

Model parameters can be tuned to maximize $g(y^*|x, M)$, where $y^*$ is the target for $x$; our objective (Eq. 1) is the expected value of the reciprocal of $g(y^*|x, M)$.

An alternative objective can be derived by first normalizing $g(y|x, M)$:

$$p(y|x, M) \equiv \frac{g(y|x, M)}{\int_{y'} g(y|x, M)} \equiv \frac{\prod_t p_t(y|x, \theta_t)}{\int_{y'} \prod_t p_t(y'|x, \theta_t) dy'}$$

This probability model underlies the product-of-experts model (Hinton, 2000) and the logarithmic opinion pool (Bordley, 1982). If we again assume $p_t(y|x, \theta_t) \sim \mathcal{N}(f_t(x), c_t^{-1})$, then $p(y|x, M)$ is a Gaussian, with mean $\overline{f(x)} = \frac{\sum_t c_t f_t(x)}{\sum_t c_t}$ and inverse variance $\bar{c} = \sum_t c_t$. The objective for this model is:

$$J^R = -\log p(y^*|x, M) = \bar{c}\left[y^* - \overline{f(x)}\right]^2 - \frac{1}{2}\log \bar{c} \qquad (2)$$

This objective corresponds to a type of residual-fitting algorithm. If $r(x) \equiv \left[y^* - \overline{f(x)}\right]$, and $\{c_t\}$ for $t < T$ are assumed frozen, then training $f_T$ to minimize $J^R$ is achieved by using $r(x)$ as a target.

These objectives can be further compared w.r.t. a bias-variance decomposition (Geman et al., 1992; Heskes, 1998). The main term in our objective can be re-expressed:

$$\sum_t c_t [y^* - f_t(x)]^2 = \sum_t c_t \left[y^* - \overline{f(x)}\right]^2 + \sum_t c_t \left[f_t(x) - \overline{f(x)}\right]^2 = \mathbf{bias} + \mathbf{variance}$$

Meanwhile, the main term of $J^R$ corresponds to the **bias** term. Hence a new hypothesis can minimize $J^R$ by having low error ($f_t(x) = y^*$), or with a deviant (ambiguous) response ($f_t(x) \neq \overline{f(x)}$) (Krogh & Vedelsby, 1995). Thus our objective attempts to minimize the average error of the models, while the residual-fitting objective minimizes the error of the average model.

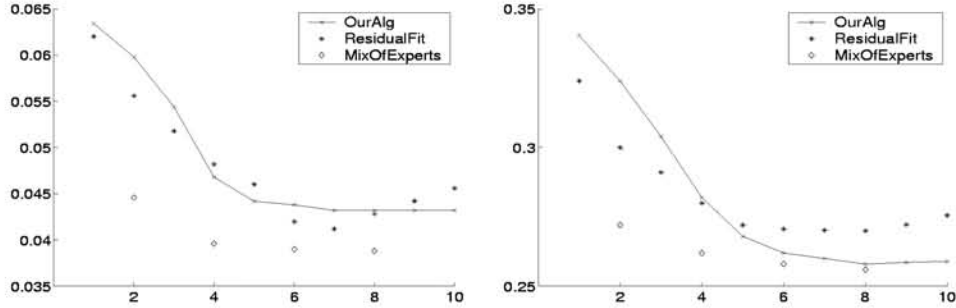

Figure 1: Generalization results for our gradient-based boosting algorithm, compared to the residual-fitting and mixture-of-experts algorithms. Left: Test problem F1; Right: Boston housing data. Normalized mean-squared error is plotted against the number of stages of boosting (or number of experts for the mixture-of-experts).

## 5   Empirical Tests of Algorithm

We report results comparing the performance of our new algorithm with two other algorithms. The first is a residual-fitting algorithm based on the $J^R$ objective (Eq. 2), but the coefficients are not normalized. The second algorithm is a version of the mixture-of-experts algorithm (Jacobs et al., 1991). Here the hypotheses (or experts) are trained simultaneously. In the standard mixture-of-experts the combination coefficients depend on the input; to make this model comparable to the others, we allowed each expert one input-independent, adaptable coefficient. This algorithm provides a good alternative to the greedy stage-wise methods, in that the experts are trained simultaneously to collectively fit the data.

We evaluate these algorithms on two problems. The first is the nonlinear prediction problem F1 (Friedman, 1991), which has 10 independent input variables uniform in $[0, 1]$: $y = 10\sin(\pi x_1 x_2) + 20(x_3 - .5)^2 + 10x_4 + 5x_5 + n$ where $n$ is a random variable drawn from a mean-zero, unit-variance normal distribution. In this problem, only five input variables ($x_1$ to $x_5$) have predictive value. We rescaled the target values $y$ to be in $[0, 3]$. We used 400 training examples, and 100 validation and test examples. The second test problem is the standard Boston Housing problem Here there are 506 examples and twelve continuous input variables. We scaled the input variables to be in $[0, 1]$, and the outputs to be in $[0, 5]$. We used 400 of the examples for training, 50 for validation, and the remainder to test generalization. We used neural networks as the hypotheses and back-propagation as the learning procedure to train them. Each network had a layer of $tanh()$ units between the input units and a single linear output. For each algorithm, we used early stopping with a validation set in order to reduce over-fitting in the hypotheses.

One finding was that the other algorithms out-performed ours when the hypotheses were simple: when the weak learners had only one or two hidden nodes, the residual-fitting algorithm reduced test error. With more hidden nodes the relative performance of our algorithm improved. Figure 1 shows average results for three-hidden-unit networks over 20 runs of each algorithm on the two problems, with examples randomly assigned to the three sets on each run. The results were consistent for different values of $\tau$ in our algorithm; here $\tau = 0.1$. Overall, the residual-fitting algorithm exhibited more over-fitting than our method. Over-fitting in these approaches may be tempered: a regularization technique known as shrinkage, which scales combination coefficients by a fractional parameter, has been found to im-

prove generalization in gradient boosting applications to classification (Friedman, 1999). Finally, the mixture-of-experts algorithm generally out-performed the sequential training algorithm. A drawback of this method is the need to specify the number of hypotheses in advance; however, given that number, simultaneous training is likely less prone to local minima than the sequential approaches.

## 6  Conclusion

We have proposed a new boosting algorithm for regression problems. Like several recent boosting methods for regression, the parameters and updates can be derived from a single common objective. Unlike these methods, our algorithm forms new hypotheses by simply modifying the distribution over training examples. Preliminary empirical comparisons have suggested that our method will not perform as well as a residual-fitting approach for simple hypotheses, but it works well for more complex ones, and it seems less prone to over-fitting. The lack of over-fitting in our method can be traced to the inherent bias-variance tradeoff, as new hypotheses are forced to resemble existing ones if they cannot improve the combined estimate.

We are exploring an extension that brings our method closer to the full mixture-of-experts. The combination coefficients can be input-dependent: a learner returns not only $f_t(\mathbf{x}^i)$ but also $k_t(\mathbf{x}^i) \in [0, 1]$, a measure of confidence in its prediction. This elaboration makes the weak learning task harder, but may extend the applicability of the algorithm: letting each learner focus on a subset of its weighted training distribution permits a divide-and-conquer approach to function approximation.

## References

[1] Bordley, R. (1982). A multiplicative formula for aggregating probability assessments. *Managment Science, 28,* 1137-1148.

[2] Breiman, L. (1997). Prediction games and arcing classifiers. TR 504. Statistics Dept., UC Berkeley.

[3] Duffy, N. & Helmbold, D. (2000). Leveraging for regression. In *Proceedings of COLT, 13.*

[4] Freund, Y. & Schapire, R. E. (1997). A decision-theoretic generalization of on-line learning and an application to boosting. *Journal of Comp. and System Sci., 55,* 119-139.

[5] Friedman, J. H. (1999). Greedy function approximation: A gradient boosting machine. TR, Dept. of Statistics, Stanford University.

[6] Friedman, J. H., Hastie, T., & Tibshirani, R. (1999). Additive logistic regression: A statistical view of boosting. *Annals of Statistics,* To appear.

[7] Geman, S., Bienenstock, E., & Doursat, R. (1992). Neural networks and the bias/variance dilemma. *Neural Computation, 4,* 1-58.

[8] Hastie, T. & Tibshirani, R. (1990). *Generalized Additive Models.* Chapman and Hall.

[9] Heskes, T. (1998). Bias-variance decompositions for likelihood-based estimators. *Neural Computation, 10,* 1425-1433.

[10] Hinton, G. E. (2000). Training products of experts by minimizing contrastive divergence. GC-NUTR 2000-004. Gatsby Computational Neuroscience Unit, University College London.

[11] Jacobs, R. A., Jordan, M. I., Nowlan, S. J., & Hinton, G. E. (1991). Adaptive mixtures of local experts. *Neural Computation, 3,* 79-87.

[12] Karakoulas, G., & Shawe-Taylor, J. (1999). Towards a strategy for boosting regressors. In *Advances in Large Margin Classifiers,* Smola, Bartlett, Schölkopf & Schuurmans (Eds.).

[13] Krogh, A. & Vedelsby, J. (1995). Neural network ensembles, cross-validation, and active learning. In *NIPS 7.*

[14] Mason, L., Baxter, J., Bartlett, P., & Frean, M. (1999). Boosting algorithms as gradient descent in function space. In *NIPS 11.*

[15] Rätsch, G., Mika, S. Onoda, T., Lemm, S. & Müller, K.-R. (2000). Barrier boosting. In *Proceedings of COLT, 13.*

[16] Schapire, R. E. (1990). The strength of weak learnability. *Machine Learning, 5,* 197-227.

[17] Schapire, R. E. & Singer, Y. (1998). Improved boosting algorithms using confidence-rated precitions. In *Proceedings of COLT, 11.*
